# Hierarchical Non-linear Factor Analysis and Topographic Maps

**Zoubin Ghahramani and Geoffrey E. Hinton**
Dept. of Computer Science, University of Toronto
Toronto, Ontario, M5S 3H5, Canada
http://www.cs.toronto.edu/neuron/
{zoubin,hinton}@cs.toronto.edu

## Abstract

We first describe a hierarchical, generative model that can be viewed as a non-linear generalisation of factor analysis and can be implemented in a neural network. The model performs perceptual inference in a probabilistically consistent manner by using top-down, bottom-up and lateral connections. These connections can be learned using simple rules that require only locally available information. We then show how to incorporate lateral connections into the generative model. The model extracts a sparse, distributed, hierarchical representation of depth from simplified random-dot stereograms and the localised disparity detectors in the first hidden layer form a topographic map. When presented with image patches from natural scenes, the model develops topographically organised local feature detectors.

## 1 Introduction

Factor analysis is a probabilistic model for real-valued data which assumes that the data is a linear combination of real-valued uncorrelated Gaussian sources (the factors). After the linear combination, each component of the data vector is also assumed to be corrupted by additional Gaussian noise. A major advantage of this generative model is that, given a data vector, the probability distribution in the space of factors is a multivariate Gaussian whose mean is a linear function of the data. It is therefore tractable to compute the posterior distribution exactly and to use it when learning the parameters of the model (the linear combination matrix and noise variances). A major disadvantage is that factor analysis is a linear model that is insensitive to higher order statistical structure of the observed data vectors.

One way to make factor analysis non-linear is to use a mixture of factor analyser modules, each of which captures a different linear regime in the data [3]. We can view the factors of *all* of the modules as a large set of basis functions for describing the data and the process of selecting one module then corresponds to selecting an appropriate subset of the basis functions. Since the number of subsets under consideration is only linear in the number of modules, it is still tractable to compute

the full posterior distribution when given a data point. Unfortunately, this mixture model is often inadequate. Consider, for example, a typical image that contains multiple objects. To represent the pose and deformation of each object we want a componential representation of the object's parameters which could be obtained from an appropriate factor analyser. But to represent the multiple objects we need several of these componential representations at once, so the pure mixture idea is not tenable. A more powerful non-linear generalisation of factor analysis is to have a large set of factors and to allow *any* subset of the factors to be selected. This can be achieved by using a generative model in which there is a high probability of generating factor activations of exactly zero.

## 2   Rectified Gaussian Belief Nets

The Rectified Gaussian Belief Net (RGBN) uses multiple layers of units with states that are either positive real values or zero [5]. Its main disadvantage is that computing the posterior distribution over the factors given a data vector involves Gibbs sampling. In general, Gibbs sampling can be very time consuming, but in practice 10 to 20 samples per unit have proved adequate and there are theoretical reasons for believing that learning can work well even when the Gibbs sampling fails to reach equilibrium [10].

We first describe the RGBN without considering neural plausibility. Then we show how lateral interactions within a layer can be used to perform probabilistic inference correctly using locally available information. This makes the RGBN far more plausible as a neural model than a sigmoid belief net [9, 8] because it means that Gibbs sampling can be performed without requiring units in one layer to see the total top-down input to units in the layer below.

The generative model for RGBN's consists of multiple layers of units each of which has a real-valued unrectified state, $y_j$, and a rectified state, $[y_j]^+$, which is zero if $y_j$ is negative and equal to $y_j$ otherwise. This rectification is the only non-linearity in the network.[1] The value of $y_j$ is Gaussian distributed with a standard deviation $\sigma_j$ and a mean, $\hat{y}_j$ that is determined by the generative bias, $g_{0j}$, and the combined effects of the rectified states of units, $k$, in the layer above:

$$\hat{y}_j = g_{0j} + \sum_k g_{kj}[y_k]^+ \tag{1}$$

The rectified state $[y_j]^+$ therefore has a Gaussian distribution above zero, but all of the mass of the Gaussian that falls below zero is concentrated in an infinitely dense spike at zero as shown in Fig. 1a. This infinite density creates problems if we attempt to use Gibbs sampling over the rectified states, so, following a suggestion by Radford Neal, we perform Gibbs sampling on the unrectified states.

Consider a unit, $j$, in some intermediate layer of a multilayer RGBN. Suppose that we fix the unrectified states of all the other units in the net. To perform Gibbs sampling, we need to stochastically select a value for $y_j$ according to its distribution given the unrectified states of all the other units. If we think in terms of energy functions, which are equal to negative log probabilities (up to a constant), the rectified states of the units in the layer above contribute a quadratic energy term by determining $\hat{y}_j$. The unrectified states of units, $i$, in the layer below contribute a constant if $[y_j]^+$ is 0, and if $[y_j]^+$ is positive they each contribute a quadratic term

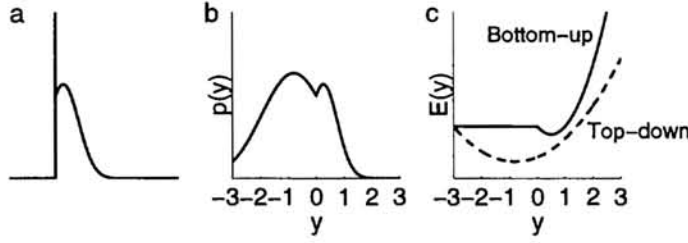

Figure 1: a) Probability density in which all the mass of a Gaussian below zero has been replaced by an infinitely dense spike at zero. b) Schematic of the density of a unit's unrectified state. c) Bottom-up and top-down energy functions corresponding to b.

because of the effect of $[y_j]^+$ on $\hat{y}_i$.

$$E(y_j) = \frac{(y_j - \hat{y}_j)^2}{2\sigma_j^2} + \sum_i \frac{(y_i - \sum_h g_{hi}[y_h]^+)^2}{2\sigma_i^2} \qquad (2)$$

where $h$ is an index over all the units in the same layer as $j$ including $j$ itself. Terms that do not depend on $y_j$ have been omitted from Eq. 2. For values of $y_j$ below zero there is a quadratic energy function which leads to a Gaussian distribution. The same is true for values of $y_j$ above zero, but it is a different quadratic (Fig. 1c). The Gaussian distributions corresponding to the two quadratics must agree at $y_j = 0$ (Fig. 1b). Because this distribution is piecewise Gaussian it is possible to perform Gibbs sampling exactly.

Given samples from the posterior, the generative weights of a RGBN can be learned by using the online delta rule to maximise the log probability of the data.[2]

$$\Delta g_{ji} = \epsilon \, [y_j]^+ \, (y_i - \hat{y}_i) \qquad (3)$$

The variance of the local Gaussian noise of each unit, $\sigma_j^2$, can also be learned by an online rule, $\Delta\sigma_j^2 = \epsilon \, [(y_j - \hat{y}_j)^2 - \sigma_j^2]$. Alternatively, $\sigma_j^2$ can be fixed at 1 for all hidden units and the effective local noise level can be controlled by scaling the generative weights.

## 3   The Role of Lateral Connections in Perceptual Inference

In RGBNs and other layered belief networks, fixing the value of a unit in one layer causes correlations between the parents of that unit in the layer above. One of the main reasons why purely bottom-up approaches to perceptual inference have proven inadequate for learning in layered belief networks is that they fail to take into account this phenomenon, which is known as "explaining away."

Lee and Seung (1997) introduced a clever way of using lateral connections to handle explaining away effects during perceptual inference. Consider the network shown in Fig. 2. One contribution, $E_{\text{below}}$, to the energy of the state of the network is the squared difference between the unrectified states of the units in one layer, $y_j$, and the top-down expectations generated by the states of units in the layer above. Assuming the local noise models for the lower layer units all have unit variance, and

ignoring biases and constant terms that are unaffected by the states of the units

$$E_{\text{below}} \;=\; \frac{1}{2}\sum_j (y_j - \hat{y}_j)^2 \;=\; \frac{1}{2}\sum_j (y_j - \sum_k [y_k]^+ g_{kj})^2. \tag{4}$$

Rearranging this expression and setting $r_{jk} = g_{kj}$ and $m_{kl} = -\sum_j g_{kj}g_{lj}$ we get

$$E_{\text{below}} = \frac{1}{2}\sum_j y_j^2 - \sum_k [y_k]^+ \sum_j y_j r_{jk} - \frac{1}{2}\sum_k [y_k]^+ \sum_l [y_l]^+ m_{kl}. \tag{5}$$

This energy function can be exactly implemented in a network with recognition weights, $r_{jk}$, and symmetric lateral interactions, $m_{kl}$. The lateral and recognition connections allow a unit, $k$, to compute how $E_{\text{below}}$ for the layer below depends on its own state and therefore they allow it to follow the gradient of $E$ or to perform Gibbs sampling in $E$.

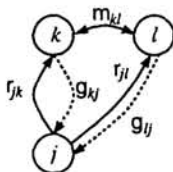

Figure 2: A small segment of a network, showing the generative weights (dashed) and the recognition and lateral weights (solid) which implement perceptual inference and correctly handle explaining away effects.

Seung's trick can be used in an RGBN and it eliminates the most neurally implausible aspect of this model which is that a unit in one layer appears to need to send both its state $y$ and the top-down prediction of its state $\hat{y}$ to units in the layer above. Using the lateral connections, the units in the layer above can, in effect, compute all they need to know about the top-down predictions. In computer simulations, we can simply set each lateral connection $m_{kl}$ to be the dot product $-\sum_j g_{kj}g_{lj}$. It is also possible to learn these lateral connections in a more biologically plausible way by driving units in the layer below with unit-variance independent Gaussian noise and using a simple anti-Hebbian learning rule. Similarly, a purely local learning rule can learn recognition weights equal to the generative weights. If units at one layer are driven by unit-variance, independent Gaussian noise, and these in turn drive units in the layer below using the generative weights, then Hebbian learning between the two layers will learn the correct recognition weights [5].

## 4   Lateral Connections in the Generative Model

When the generative model contains only top-down connections, lateral connections make it possible to do perceptual inference using locally available information. But it is also possible, and often desirable, to have lateral connections in the generative model. Such connections can cause nearby units in a layer to have *a priori* correlated activities, which in turn can lead to the formation of redundant codes and, as we will see, topographic maps.

Symmetric lateral interactions between the unrectified states of units within a layer have the effect of adding a quadratic term to the energy function

$$E_{\text{MRF}} = \frac{1}{2}\sum_k \sum_l M_{kl}\, y_k y_l, \tag{6}$$

which corresponds to a Gaussian Markov Random Field (MRF). During sampling, this term is simply added to the top-down energy contribution. Learning is more difficult. The difficulty stems from the need to know the derivatives of the partition function of the MRF for each data vector. This partition function depends on the

top-down inputs to a layer so it varies from one data vector to the next, even if the lateral connections themselves are non-adaptive. Fortunately, since both the MRF and the top-down prediction define Gaussians over the states of the units in a layer, these derivatives can be easily calculated. Assuming unit variances,

$$\Delta g_{ji} = \epsilon \left( [y_j]^+ (y_i - \hat{y}_i) + [y_j]^+ \sum_k \left[ M(I+M)^{-1} \right]_{ik} \hat{y}_k \right) \qquad (7)$$

where $M$ is the MRF matrix for the layer including units $i$ and $k$, and $I$ is the identity matrix. The first term is the delta rule (Eq. 3); the second term is the derivative of the partition function which unfortunately involves a matrix inversion. Since the partition function for a multivariate Gaussian is analytical it is also possible to learn the lateral connections in the MRF.

Lateral interactions between the *rectified* states of units add the quadratic term $\frac{1}{2} \sum_k \sum_l M_{kl} [y_k]^+ [y_l]^+$. The partition function is no longer analytical, so computing the gradient of the likelihood involves a two-phase Boltzmann-like procedure:

$$\Delta g_{ji} = \epsilon \left( \langle [y_j]^+ y_i \rangle^* - \langle [y_j]^+ y_i \rangle^- \right), \qquad (8)$$

where $\langle \cdot \rangle^*$ averages with respect to the posterior distribution of $y_i$ and $y_j$, and $\langle \cdot \rangle^-$ averages with respect to the posterior distribution of $y_j$ and the prior of $y_i$ given units in the same layer as $j$. This learning rule suffers from all the problems of the Boltzmann machine, namely it is slow and requires two-phases. However, there is an approximation which results in the familiar one-phase delta rule that can be described in three equivalent ways: (1) it treats the lateral connections in the generative model as if they were additional lateral connections in the recognition model; (2) instead of lateral connections in the generative model it assumes some fictitious children with clamped values which affect inference but whose likelihood is not maximised during learning; (3) it maximises a penalized likelihood of the model without the lateral connections in the generative model.

## 5   Discovering depth in simplified stereograms

Consider the following generative process for stereo pairs. Random dots of uniformly distributed intensities are scattered sparsely on a one-dimensional surface, and the image is blurred with a Gaussian filter. This surface is then randomly placed at one of two different depths, giving rise to two possible left-to-right disparities between the images seen by each eye. Separate Gaussian noise is then added to the image seen by each eye. Some images generated in this manner are shown in Fig. 3a.

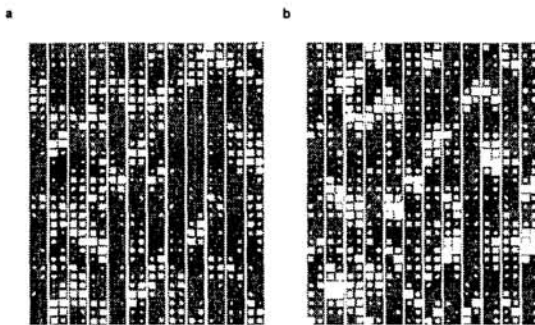

Figure 3: **a)** Sample data from the stereo disparity problem. The left and right column of each $2 \times 32$ image are the inputs to the left and right eye, respectively. Periodic boundary conditions were used. The value of a pixel is represented by the size of the square, with white being positive and black being negative. Notice that pixel noise makes it difficult to infer the disparity, *i.e.* the vertical shift between the left and right columns, in some images. **b)** Sample images generated by the model after learning.

We trained a three-layer RGBN consisting of 64 visible units, 64 units in the first hidden layer and 1 unit in the second hidden layer on the 32-pixel wide stereo

disparity problem. Each of the hidden units in the first hidden layer was connected to the entire array of visible units, *i.e.* it had inputs from both eyes. The hidden units in this layer were also laterally connected in an MRF over the unrectified units. Nearby units excited each other and more distant units inhibited each other, with the net pattern of excitation/inhibition being a difference of two Gaussians. This MRF was initialised with large weights which decayed exponentially to zero over the course of training. The network was trained for 30 passes through a data set of 2000 images. For each image we used 16 iterations of Gibbs sampling to approximate the posterior distribution over hidden states. Each iteration consisted of sampling every hidden unit once in a random order. The states after the fourth iteration of Gibbs sampling were used for learning, with a learning rate of 0.05 and a weight decay parameter of 0.001. Since the top level of the generative process makes a discrete decision between left and right global disparity we used a trivial extension of the RGBN in which the top level unit saturates both at 0 and 1.

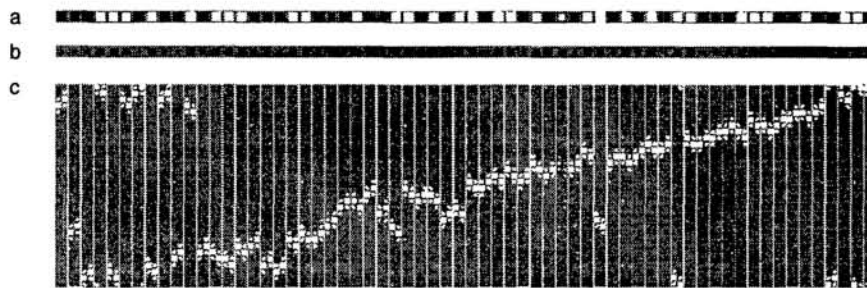

Figure 4: Generative weights of a three-layered RGBN after being trained on the stereo disparity problem. **a)** Weights from the top layer hidden unit to the 64 middle-layer hidden units. **b)** Biases of the middle-layer hidden units, and **c)** weights from the hidden units to the $2 \times 32$ visible array.

Thirty-two of the hidden units learned to become local left-disparity detectors, while the other 32 became local right-disparity detectors (Fig. 4c). The unit in the second hidden layer learned positive weights to the left-disparity detectors in the layer below, and negative weights to the right detectors (Fig. 4a). In fact, the activity of this top unit discriminated the true global disparity of the input images with 99% accuracy. A random sample of images generated by the model after learning is shown in Fig. 3b. In addition to forming a hierarchical distributed representation of disparity, units in the hidden layer self-organised into a topographic map. The MRF caused high correlations between nearby units early in learning, which in turn resulted in nearby units learning similar weight vectors. The emergence of topography depended on the strength of the MRF and on the speed with which it decayed. Results were relatively insensitive to other parametric changes.

We also presented image patches taken from natural images [1] to a network with units in the first hidden layer arranged in laterally-connected 2D grid. The network developed local feature detectors, with nearby units responding to similar features (Fig. 5). Not all units were used, but the unused units all clustered into one area.

## 6   Discussion

Classical models of topography formation such as Kohonen's self-organising map [6] and the elastic net [2, 4] can be thought of as variations on mixture models where additional constraints have been placed to encourage neighboring hidden units to have similar generative weights. The problem with a mixture model is that it cannot handle images in which there are several things going on at once. In contrast, we

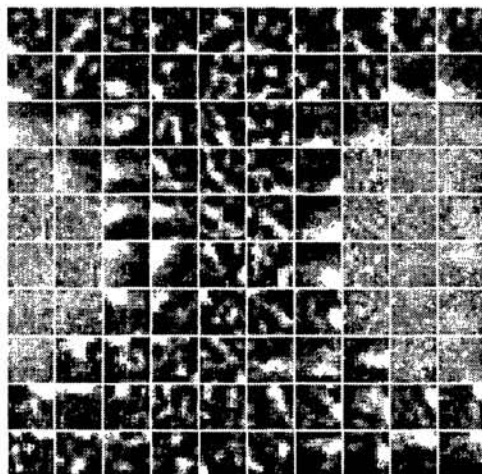

Figure 5: Generative weights of an RGBN trained on 12 × 12 natural image patches: weights from each of the 100 hidden units which were arranged in a 10 × 10 sheet with toroidal boundary conditions.

have shown that topography can arise in much richer hierarchical and componential generative models by inducing correlations between neighboring units.

There is a sense in which topography is a necessary consequence of the lateral connection trick used for perceptual inference. It is infeasible to interconnect all pairs of units in a cortical area. If we assume that direct lateral interactions (or interactions mediated by interneurons) are primarily local, then widely separated units will not have the apparatus required for explaining away. Consequently the computation of the posterior distribution will be incorrect unless the generative weight vectors of widely separated units are orthogonal. If the generative weights are constrained to be positive, the only way two vectors can be orthogonal is for each to have zeros wherever the other has non-zeros. Since the redundancies that the hidden units are trying to model are typically spatially localised, it follows that widely separated units must attend to different parts of the image and units can only attend to overlapping patches if they are laterally interconnected. The lateral connections in the generative model assist in the formation of the topography required for correct perceptual inference.

**Acknowledgements.** We thank P. Dayan, B. Frey, G. Goodhill, D. MacKay, R. Neal and M. Revow. The research was funded by NSERC and ITRC. GEH is the Nesbitt-Burns fellow of CIAR.

## Footnotes

[1]The key arguments presented in this paper hold for general nonlinear belief networks as long as the noise is Gaussian; they are not specific to the rectification nonlinearity.

[2]If Gibbs sampling has not been run long enough to reach equilibrium, the delta rule follows the gradient of the *penalized* log probability of the data [10]. The penalty term is the Kullback-Liebler divergence between the equilibrium distribution and the distribution produced by Gibbs sampling. Other things being equal, the delta rule therefore adjusts the parameters that determine the equilibrium distribution to reduce this penalty, thus favouring models for which Gibbs sampling works quickly.

# References

[1] A. Bell & T. J. Sejnowski. The 'Independent components' of natural scenes are edge filters. *Vision Research*, In Press.

[2] R. Durbin & D. Willshaw. An analogue approach to the travelling salesman problem using an elastic net method. *Nature*, 326(16):689–691, 1987.

[3] Z. Ghahramani & G. E. Hinton. The EM algorithm for mixtures of factor analyzers. Univ. Toronto Technical Report CRG-TR-96-1, 1996.

[4] G. J. Goodhill & D. J. Willshaw. Application of the elatic net algorithm to the formation of ocular dominance stripes. *Network: Comp. in Neur. Sys.*, 1:41–59, 1990.

[5] G. E. Hinton & Z. Ghahramani. Generative models for discovering sparse distributed representations. *Philos. Trans. Roy. Soc. B*, 352:1177–1190, 1997.

[6] T. Kohonen. Self-organized formation of topologically correct feature maps. *Biological Cybernetics*, 43:59–69, 1982.

[7] D. D. Lee & H. S. Seung. Unsupervised learning by convex and conic coding. In M. Mozer, M. Jordan, & T. Petsche, eds., *NIPS 9*. MIT Press, Cambridge, MA, 1997.

[8] M. S. Lewicki & T. J. Sejnowski. Bayesian unsupervised learning of higher order structure. In *NIPS 9*. MIT Press, Cambridge, MA, 1997.

[9] R. M. Neal. Connectionist learning of belief networks. *Artif. Intell.*, 56:71–113, 1992.

[10] R. M. Neal & G. E. Hinton. A new view of the EM algorithm that justifies incremental and other variants. *Unpublished Manuscript*, 1993.